# Bayesian PCA

**Christopher M. Bishop**

Microsoft Research
St. George House, 1 Guildhall Street
Cambridge CB2 3NH, U.K.
cmbishop@microsoft.com

## Abstract

The technique of principal component analysis (PCA) has recently been expressed as the maximum likelihood solution for a generative latent variable model. In this paper we use this probabilistic reformulation as the basis for a Bayesian treatment of PCA. Our key result is that *effective* dimensionality of the latent space (equivalent to the number of retained principal components) can be determined automatically as part of the Bayesian inference procedure. An important application of this framework is to mixtures of probabilistic PCA models, in which each component can determine its own effective complexity.

## 1 Introduction

Principal component analysis (PCA) is a widely used technique for data analysis. Recently Tipping and Bishop (1997b) showed that a specific form of generative latent variable model has the property that its maximum likelihood solution extracts the principal sub-space of the observed data set. This probabilistic reformulation of PCA permits many extensions including a principled formulation of mixtures of principal component analyzers, as discussed by Tipping and Bishop (1997a).

A central issue in maximum likelihood (as well as conventional) PCA is the choice of the number of principal components to be retained. This is particularly problematic in a mixture modelling context since ideally we would like the components to have potentially different dimensionalities. However, an exhaustive search over the choice of dimensionality for each of the components in a mixture distribution can quickly become computationally intractable. In this paper we develop a Bayesian treatment of PCA, and we show how this leads to an *automatic* selection of the appropriate model dimensionality. Our approach avoids a discrete model search, involving instead the use of continuous hyper-parameters to determine an *effective* number of principal components.

## 2 Maximum Likelihood PCA

Consider a data set $D$ of observed $d$-dimensional vectors $D = \{\mathbf{t}_n\}$ where $n \in \{1, \ldots, N\}$. Conventional principal component analysis is obtained by first computing the sample covariance matrix given by

$$\mathbf{S} = \frac{1}{N} \sum_{n=1}^{N} (\mathbf{t}_n - \bar{\mathbf{t}})(\mathbf{t}_n - \bar{\mathbf{t}})^{\mathrm{T}} \tag{1}$$

where $\bar{\mathbf{t}} = N^{-1} \sum_n \mathbf{t}_n$ is the sample mean. Next the eigenvectors $\mathbf{u}_i$ and eigenvalues $\lambda_i$ of $\mathbf{S}$ are found, where $\mathbf{S}\mathbf{u}_i = \lambda_i \mathbf{u}_i$ and $i = 1, \ldots, d$. The eigenvectors corresponding to the $q$ largest eigenvalues (where $q < d$) are retained, and a reduced-dimensionality representation of the data set is defined by $\mathbf{x}_n = \mathbf{U}^{\mathrm{T}}(\mathbf{t}_n - \bar{\mathbf{t}})$ where $\mathbf{U}_q = (\mathbf{u}_1, \ldots, \mathbf{u}_q)$. It is easily shown that PCA corresponds to the linear projection of a data set under which the retained variance is a maximum, or equivalently the linear projection for which the sum-of-squares reconstruction cost is minimized.

A significant limitation of conventional PCA is that it does not define a probability distribution. Recently, however, Tipping and Bishop (1997b) showed how PCA can be reformulated as the maximum likelihood solution of a specific latent variable model, as follows. We first introduce a $q$-dimensional latent variable $\mathbf{x}$ whose prior distribution is a zero mean Gaussian $p(\mathbf{x}) = \mathcal{N}(\mathbf{0}, \mathbf{I}_q)$ and $\mathbf{I}_q$ is the $q$-dimensional unit matrix. The observed variable $\mathbf{t}$ is then defined as a linear transformation of $\mathbf{x}$ with additive Gaussian noise $\mathbf{t} = \mathbf{W}\mathbf{x} + \boldsymbol{\mu} + \boldsymbol{\epsilon}$ where $\mathbf{W}$ is a $d \times q$ matrix, $\boldsymbol{\mu}$ is a $d$-dimensional vector and $\boldsymbol{\epsilon}$ is a zero-mean Gaussian-distributed vector with covariance $\sigma^2 \mathbf{I}_d$. Thus $p(\mathbf{t}|\mathbf{x}) = \mathcal{N}(\mathbf{W}\mathbf{x} + \boldsymbol{\mu}, \sigma^2 \mathbf{I}_d)$. The marginal distribution of the observed variable is then given by the convolution of two Gaussians and is itself Gaussian

$$p(\mathbf{t}) = \int p(\mathbf{t}|\mathbf{x})p(\mathbf{x}) \, d\mathbf{x} = \mathcal{N}(\boldsymbol{\mu}, \mathbf{C}) \tag{2}$$

where the covariance matrix $\mathbf{C} = \mathbf{W}\mathbf{W}^{\mathrm{T}} + \sigma^2 \mathbf{I}_d$. The model (2) represents a constrained Gaussian distribution governed by the parameters $\boldsymbol{\mu}$, $\mathbf{W}$ and $\sigma^2$.

The log probability of the parameters under the observed data set $D$ is then given by

$$L(\boldsymbol{\mu}, \mathbf{W}, \sigma^2) = -\frac{N}{2} \left\{ d\ln(2\pi) + \ln|\mathbf{C}| + \mathrm{Tr}\left[\mathbf{C}^{-1}\mathbf{S}\right] \right\} \tag{3}$$

where $\mathbf{S}$ is the sample covariance matrix given by (1). The maximum likelihood solution for $\boldsymbol{\mu}$ is easily seen to be $\boldsymbol{\mu}_{\mathrm{ML}} = \bar{\mathbf{t}}$. It was shown by Tipping and Bishop (1997b) that the stationary points of the log likelihood with respect to $\mathbf{W}$ satisfy

$$\mathbf{W}_{\mathrm{ML}} = \mathbf{U}_q (\boldsymbol{\Lambda}_q - \sigma^2 \mathbf{I}_q)^{1/2} \tag{4}$$

where the columns of $\mathbf{U}_q$ are eigenvectors of $\mathbf{S}$, with corresponding eigenvalues in the diagonal matrix $\boldsymbol{\Lambda}_q$. It was also shown that the *maximum* of the likelihood is achieved when the $q$ largest eigenvalues are chosen, so that the columns of $\mathbf{U}_q$ correspond to the *principal* eigenvectors, with all other choices of eigenvalues corresponding to saddle points. The maximum likelihood solution for $\sigma^2$ is then given by

$$\sigma^2_{\mathrm{ML}} = \frac{1}{d-q} \sum_{i=q+1}^{d} \lambda_i \tag{5}$$

which has a natural interpretation as the average variance lost per discarded dimension. The density model (2) thus represents a probabilistic formulation of PCA. It is easily verified that conventional PCA is recovered in the limit $\sigma^2 \to 0$.

Probabilistic PCA has been successfully applied to problems in data compression, density estimation and data visualization, and has been extended to mixture and hierarchical mixture models. As with conventional PCA, however, the model itself provides no mechanism for determining the value of the latent-space dimensionality $q$. For $q = d - 1$ the model is equivalent to a full-covariance Gaussian distribution, while for $q < d - 1$ it represents a constrained Gaussian in which the variance in the remaining $d - q$ directions is modelled by the single parameter $\sigma^2$. Thus the choice of $q$ corresponds to a problem in model complexity optimization. If data is plentiful, then cross-validation to compare all possible values of $q$ offers a possible approach. However, this can quickly become intractable for mixtures of probabilistic PCA models if we wish to allow each component to have its own $q$ value.

## 3  Bayesian PCA

The issue of model complexity can be handled naturally within a Bayesian paradigm. Armed with the probabilistic reformulation of PCA defined in Section 2, a Bayesian treatment of PCA is obtained by first introducing a prior distribution $p(\mu, \mathbf{W}, \sigma^2)$ over the parameters of the model. The corresponding posterior distribution $p(\mu, \mathbf{W}, \sigma^2 | D)$ is then obtained by multiplying the prior by the likelihood function, whose logarithm is given by (3), and normalizing. Finally, the predictive density is obtained by marginalizing over the parameters, so that

$$p(\mathbf{t}|D) = \iiint p(\mathbf{t}|\mu, \mathbf{W}, \sigma^2) p(\mu, \mathbf{W}, \sigma^2 | D) \, d\mu \, d\mathbf{W} \, d\sigma^2. \tag{6}$$

In order to implement this framework we must address two issues: (i) the choice of prior distribution, and (ii) the formulation of a tractable algorithm. Our focus in this paper is on the specific issue of controlling the effective dimensionality of the latent space (corresponding to the number of retained principal components). Furthermore, we seek to avoid discrete model selection and instead use continuous hyper-parameters to determine automatically an appropriate *effective* dimensionality for the latent space as part of the process of Bayesian inference. This is achieved by introducing a *hierarchical* prior $p(\mathbf{W}|\boldsymbol{\alpha})$ over the matrix $\mathbf{W}$, governed by a $q$-dimensional vector of hyper-parameters $\boldsymbol{\alpha} = \{\alpha_1, \ldots, \alpha_q\}$. The dimensionality of the latent space is set to its maximum possible value $q = d - 1$, and each hyper-parameter controls one of the columns of the matrix $\mathbf{W}$ through a conditional Gaussian distribution of the form

$$p(\mathbf{W}|\boldsymbol{\alpha}) = \prod_{i=1}^{d-1} \left(\frac{\alpha_i}{2\pi}\right)^{d/2} \exp\left\{-\frac{1}{2}\alpha_i \|\mathbf{w}_i\|^2\right\} \tag{7}$$

where $\{\mathbf{w}_i\}$ are the columns of $\mathbf{W}$. This form of prior is motivated by the framework of *automatic relevance determination* (ARD) introduced in the context of neural networks by Neal and MacKay (see MacKay, 1995). Each $\alpha_i$ controls the inverse variance of the corresponding $\mathbf{w}_i$, so that if a particular $\alpha_i$ has a posterior distribution concentrated at large values, the corresponding $\mathbf{w}_i$ will tend to be small, and that direction in latent space will be effectively 'switched off'. The probabilistic structure of the model is displayed graphically in Figure 1.

In order to make use of this model in practice we must be able to marginalize over the posterior distribution of $\mathbf{W}$. Since this is analytically intractable we have developed three alternative approaches based on (i) type-II maximum likelihood using a local Gaussian approximation to a mode of the posterior distribution (MacKay, 1995), (ii) Markov chain Monte Carlo using Gibbs sampling, and (iii) variational inference using a factorized approximation to the posterior distribution. Here we describe the first of these in more detail.

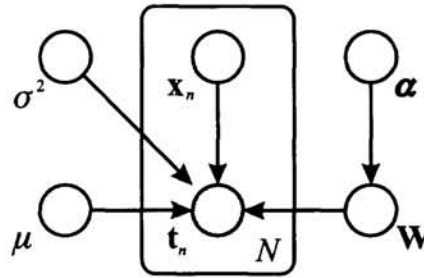

Figure 1: Representation of Bayesian PCA as a probabilistic graphical model showing the hierarchical prior over $\mathbf{W}$ governed by the vector of hyper-parameters $\boldsymbol{\alpha}$. The box denotes a 'plate' comprising a data set of $N$ independent observations of the visible vector $\mathbf{t}_n$ (shown shaded) together with the corresponding hidden variables $\mathbf{x}_n$.

The location $\mathbf{W}_{\mathrm{MP}}$ of the mode can be found by maximizing the log posterior distribution given, from Bayes' theorem, by

$$\ln p(\mathbf{W}|D) = L - \frac{1}{2}\sum_{i=1}^{d-1}\alpha_i\|\mathbf{w}_i\|^2 + \text{const.} \tag{8}$$

where $L$ is given by (3). For the purpose of controlling the effective dimensionality of the latent space, it is sufficient to treat $\boldsymbol{\mu}$, $\sigma^2$ and $\boldsymbol{\alpha}$ as parameters whose values are to be estimated, rather than as random variables. In this case there is no need to introduce priors over these variables, and we can determine $\boldsymbol{\mu}$ and $\sigma^2$ by maximum likelihood. To estimate $\boldsymbol{\alpha}$ we use type-II maximum likelihood, corresponding to maximizing the marginal likelihood $p(D|\boldsymbol{\alpha})$ in which we have integrated over $\mathbf{W}$ using the quadratic approximation. It is easily shown (Bishop, 1995) that this leads to a re-estimation formula for the hyper-parameters $\alpha_i$ of the form

$$\alpha_i := \frac{\gamma_i}{\|\mathbf{w}_i\|^2} \tag{9}$$

where $\gamma_i = d - \alpha_i\mathrm{Tr}_i(\mathbf{H}^{-1})$ is the effective number of parameters in $\mathbf{w}_i$, $\mathbf{H}$ is the Hessian matrix given by the second derivatives of $\ln p(\mathbf{W}|D)$ with respect to the elements of $\mathbf{W}$ (evaluated at $\mathbf{W}_{\mathrm{MP}}$), and $\mathrm{Tr}_i(\cdot)$ denotes the trace of the sub-matrix corresponding to the vector $\mathbf{w}_i$.

For the results presented in this paper, we make the further simplification of replacing $\gamma_i$ in (9) by $d$, corresponding to the assumption that all model parameters are 'well-determined'. This significantly reduces the computational cost since it avoids evaluation and manipulation of the Hessian matrix. An additional consequence is that vectors $\mathbf{w}_i$ for which there is insufficient support from the data will be driven to zero, with the corresponding $\alpha_i \to \infty$, so that un-used dimensions are switched off completely. We define the *effective* dimensionality of the model to be the number of vectors $\mathbf{w}_i$ whose values remain non-zero.

The solution for $\mathbf{W}_{\mathrm{MP}}$ can be found efficiently using the EM algorithm, in which the E-step involves evaluation of the expected sufficient statistics of the latent-space posterior distribution, given by

$$\langle\mathbf{x}_n\rangle = \mathbf{M}^{-1}\mathbf{W}^{\mathrm{T}}(\mathbf{t}_n - \boldsymbol{\mu}) \tag{10}$$

$$\langle\mathbf{x}_n\mathbf{x}_n^{\mathrm{T}}\rangle = \sigma^2\mathbf{M} + \langle\mathbf{x}_n\rangle\langle\mathbf{x}_n\rangle^{\mathrm{T}} \tag{11}$$

where $\mathbf{M} = (\mathbf{W}^T\mathbf{W} + \sigma^2\mathbf{I}_q)$. The M-step involves updating the model parameters using

$$\widetilde{\mathbf{W}} \; = \; \left[\sum_n (\mathbf{t}_n - \boldsymbol{\mu})\langle \mathbf{x}_n^T \rangle\right]\left[\sum_n \langle \mathbf{x}_n\mathbf{x}_n^T \rangle + \sigma^2\mathbf{A}\right]^{-1} \tag{12}$$

$$\widetilde{\sigma}^2 \; = \; \frac{1}{Nd}\sum_{n=1}^{N}\left\{\|\mathbf{t}_n - \boldsymbol{\mu}\|^2 - 2\langle \mathbf{x}_n^T\rangle\widetilde{\mathbf{W}}^T(\mathbf{t}_n - \boldsymbol{\mu}) + \mathrm{Tr}\left[\langle\mathbf{x}_n\mathbf{x}_n^T\rangle\widetilde{\mathbf{W}}^T\widetilde{\mathbf{W}}\right]\right\} \tag{13}$$

where $\mathbf{A} = \mathrm{diag}(\alpha_i)$. Optimization of $\mathbf{W}$ and $\sigma^2$ is alternated with re-estimation of $\boldsymbol{\alpha}$, using (9) with $\gamma_i = d$, until all of the parameters satisfy a suitable convergence criterion.

As an illustration of the operation of this algorithm, we consider a data set consisting of 300 points in 10 dimensions, in which the data is drawn from a Gaussian distribution having standard deviation 1.0 in 3 directions and standard deviation 0.5 in the remaining 7 directions. The result of fitting both maximum likelihood and Bayesian PCA models is shown in Figure 2. In this case the Bayesian model has an effective dimensionality of $q_{\mathrm{eff}} = 3$.

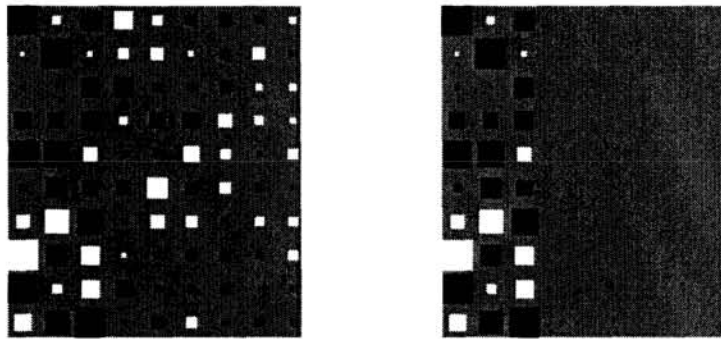

Figure 2: Hinton diagrams of the matrix $\mathbf{W}$ for a data set in 10 dimensions having $m = 3$ directions with larger variance than the remaining 7 directions. The left plot shows $\mathbf{W}$ from maximum likelihood PCA while the right plot shows $\mathbf{W}_{\mathrm{MP}}$ from the Bayesian approach, showing how the model is able to discover the appropriate dimensionality by suppressing the 6 surplus degrees of freedom.

The effective dimensionality found by Bayesian PCA will be dependent on the number $N$ of points in the data set. For $N \to \infty$ we expect $q_{\mathrm{eff}} \to d-1$, and in this limit the maximum likelihood framework and the Bayesian approach will give identical results. For finite data sets the effective dimensionality may be reduced, with degrees of freedom for which there is insufficient evidence in the data set being suppressed. The variance of the data in the remaining $d - q_{\mathrm{eff}}$ directions is then accounted for by the single degree of freedom defined by $\sigma^2$. This is illustrated by considering data in 10 dimensions generated from a Gaussian distribution with standard deviations given by $\{1.0, 0.9, 0.8, 0.7, 0.6, 0.5, 0.4, 0.3, 0.2, 0.1\}$. In Figure 3 we plot $q_{\mathrm{eff}}$ (averaged over 50 independent experiments) versus the number $N$ of points in the data set.

These results indicate that Bayesian PCA is able to determine automatically a suitable effective dimensionality $q_{\mathrm{eff}}$ for the principal component subspace, and therefore offers a practical alternative to exhaustive comparison of dimensionalities using techniques such as cross-validation. As an illustration of the generalization capability of the resulting model we consider a data set of 20 points in 10 dimensions generated from a Gaussian distribution having standard deviations in 5 directions given by $(1.0, 0.8, 0.6, 0.4, 0.2)$ and standard deviation 0.04 in the remaining 5 directions. We fit maximum likelihood PCA models to this data having $q$ values in the range 1–9 and compare their log likelihoods on both the training data and on an independent test set, with the results (averaged over 10 independent experiments) shown in Figure 4. Also shown are the corresponding results obtained from Bayesian PCA.

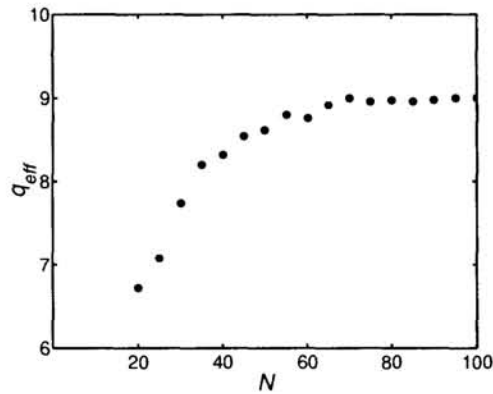

Figure 3: Plot of the average effective dimensionality of the Bayesian PCA model versus the number $N$ of data points for data in a 10-dimensional space.

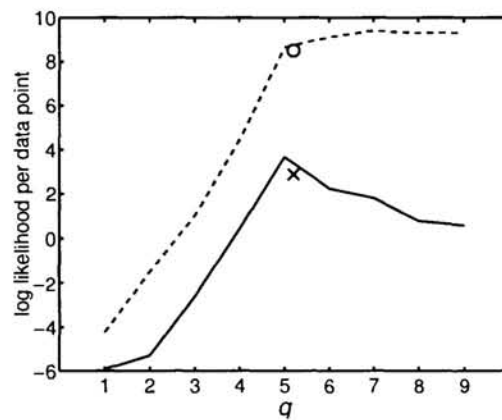

Figure 4: Plot of the log likelihood for the training set (dashed curve) and the test set (solid curve) for maximum likelihood PCA models having $q$ values in the range 1–9, showing that the best generalization is achieved for $q = 5$ which corresponds to the number of directions of significant variance in the data set. Also shown are the training (circle) and test (cross) results from a Bayesian PCA model, plotted at the average effective $q$ value given by $q_{\text{eff}} = 5.2$. We see that the Bayesian PCA model automatically discovers the appropriate dimensionality for the principal component subspace, and furthermore that it has a generalization performance which is close to that of the optimal fixed $q$ model.

## 4   Mixtures of Bayesian PCA Models

Given a probabilistic formulation of PCA it is straightforward to construct a mixture distribution comprising a linear superposition of principal component analyzers. In the case of maximum likelihood PCA we have to choose both the number $M$ of components and the latent space dimensionality $q$ for each component. For moderate numbers of components and data spaces of several dimensions it quickly becomes intractable to explore the exponentially large number of combinations of $q$ values for a given value of $M$. Here Bayesian PCA offers a significant advantage in allowing the effective dimensionalities of the models to be determined automatically.

As an illustration we consider a density estimation problem involving hand-written digits from the CEDAR database. The data set comprises $8 \times 8$ scaled and smoothed gray-scale images of the digits '2', '3' and '4', partitioned randomly into 1500 training, 900 validation and 900 test points. For mixtures of maximum likelihood PCA the model parameters can be

determined using the EM algorithm in which the M-step uses (4) and (5), with eigenvector and eigenvalues obtained from the weighted covariance matrices in which the weighting coefficients are the posterior probabilities for the components determined in the E-step. Since, for maximum likelihood PCA, it is computationally impractical to explore independent $q$ values for each component we consider mixtures in which every component has the same dimensionality. We therefore train mixtures having $M \in \{2, 4, 6, 8, 10, 12, 14, 16, 18\}$ for all values $q \in \{2, 4, 8, 12, 16, 20, 25, 30, 40, 50\}$. In order to avoid singularities associated with the more complex models we omit any component from the mixture for which the value of $\sigma^2$ goes to zero during the optimization. The highest log likelihood on the validation set ($-295$) is obtained for $M = 6$ and $q = 50$.

For mixtures of Bayesian PCA models we need only explore alternative values for $M$, which are taken from the same set as for the mixtures of maximum likelihood PCA. Again, the best performance on the validation set ($-293$) is obtained for $M = 6$. The values of the log likelihood for the test set were $-295$ (maximum likelihood PCA) and $-293$ (Bayesian PCA). The mean vectors $\mu_i$ for each of the 6 components of the Bayesian PCA mixture model are shown in Figure 5.

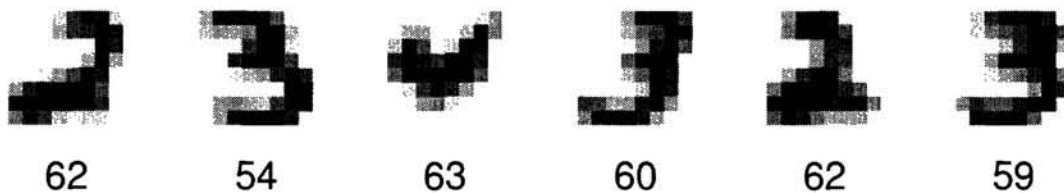

<div align="center">
62     54     63     60     62     59
</div>

Figure 5: The mean vectors for each of the 6 components in the Bayesian PCA mixture model, displayed as an $8 \times 8$ image, together with the corresponding values of the effective dimensionality.

The Bayesian treatment of PCA discussed in this paper can be particularly advantageous for small data sets in high dimensions as it can avoid the singularities associated with maximum likelihood (or conventional) PCA by suppressing unwanted degrees of freedom in the model. This is especially helpful in a mixture modelling context, since the effective number of data points associated with specific 'clusters' can be small even when the total number of data points appears to be large.

# References

Bishop, C. M. (1995). *Neural Networks for Pattern Recognition*. Oxford University Press.

MacKay, D. J. C. (1995). Probable networks and plausible predictions – a review of practical Bayesian methods for supervised neural networks. *Network: Computation in Neural Systems* **6** (3), 469–505.

Tipping, M. E. and C. M. Bishop (1997a). Mixtures of principal component analysers. In *Proceedings IEE Fifth International Conference on Artificial Neural Networks, Cambridge, U.K., July.*, pp. 13–18.

Tipping, M. E. and C. M. Bishop (1997b). Probabilistic principal component analysis. Accepted for publication in the Journal of the Royal Statistical Society, B.
